# Pattern Playback in the '90s

**Malcolm Slaney**
Interval Research Corporation
1801-C Page Mill Road,
Palo Alto, CA 94304
malcolm@interval.com

## Abstract

Deciding the appropriate representation to use for modeling human auditory processing is a critical issue in auditory science. While engineers have successfully performed many single-speaker tasks with LPC and spectrogram methods, more difficult problems will need a richer representation. This paper describes a powerful auditory representation known as the correlogram and shows how this non-linear representation can be converted back into sound, with no loss of perceptually important information. The correlogram is interesting because it is a neurophysiologically plausible representation of sound. This paper shows improved methods for spectrogram inversion (conventional pattern playback), inversion of a cochlear model, and inversion of the correlogram representation.

## 1 INTRODUCTION[1]

My interest in auditory models and perceptual displays [2] is motivated by the problem of sound understanding, especially the separation of speech from noisy backgrounds and interfering speakers. The correlogram and related representations are a pattern space within which sounds can be "understood" and "separated" [3][4]. I am therefore interested in resynthesizing sounds from these representations as a way to test and evaluate sound separation algorithms, and as a way to apply sound separation to problems such as speech enhancement. The conversion of sound to a correlogram involves the intermediate representation of a cochleagram, as shown in Figure 1, so cochlear-model inversion is addressed as one piece of the overall problem.

---

1. Much of this work was performed by Malcolm Slaney, Daniel Naar and Richard F. Lyon while all three were employed at Apple Computer. The mathematical details of this work were presented at the 1994 ICASSP[1].

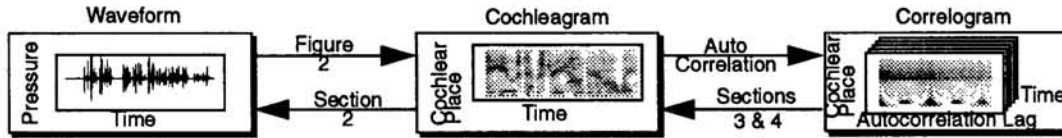

Figure 1. Three stages in low-level auditory perception are shown here. Sound waves are converted into a detailed representation with broad spectral bands, known as cochleagrams. The correlogram then summarizes the periodicities in the cochleagram with short-time autocorrelation. The result is a perceptual movie synchronized to the acoustic signal. The two inversion problems addressed in this work are indicated with arrows from right to left.

There are three factors which can be used to judge the quality of an auditory model: psychoacoustic comparisons, neurophysiological plausibility, and does it represent the perceptually relevant information? First, the correlogram has been shown to simply and accurately predict human pitch perception [5]. The neurophysiological basis for the correlogram has not been found, but there are neural circuits performing the same calculation in the mustached bat's echolocation system [6]. Finally, from an information representation point of view, does the correlogram preserve the salient information? The results of this paper show that no information has been lost. Since the psychoacoustic, neurophysiological, and information representation measures are all positive, perhaps the correlogram is the basis of most auditory processing.

The inversion techniques described here are important because they allow us to readily evaluate the results of sound separation models that "zero out" unwanted portions of the signal in the correlogram domain. This work extends the convex projection approach of Irino [7] and Yang [8] by considering a different cochlear model, and by including the correlogram inversion. The convex projection approach is well suited to "filling in" missing information. While this paper only describes the process for one particular auditory model, the techniques are equally useful for other models.

This paper describes three aspects of the problem: cochleagram inversion, conversion of the correlogram into spectrograms, and spectrogram inversion. A number of reconstruction options are explored in this paper. Some are fast, while other techniques use time-consuming iterations to produce reconstructions perceptually equivalent to the original sound. Fast versions of these algorithms could allow us to separate a speaker's voice from the background noise in real time.

## 2  COCHLEAGRAM INVERSION

Figure 2 shows a block diagram of the cochlear model [9] that is used in this work. The basis of the model is a bank of filters, implemented as a cascade of low-pass filters, that splits the input signal into broad spectral bands. The output from each filter in the bank is called a channel. The energy in each channel is detected and used to adjust the channel gain, implementing a simple model of auditory sensitivity adaptation, or automatic gain control (AGC). The half-wave rectifier (HWR) detection nonlinearity provides a waveform for each channel that roughly represents the instantaneous neural firing rate at each position along the cochlea.

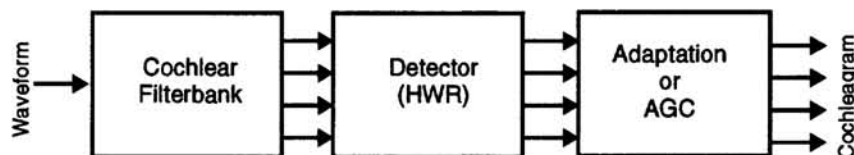

Figure 2. Three stages of the simple cochlear model used in this paper are shown above.

The cochleagram is converted back into sound by reversing the three steps shown in Figure 2. First the AGC is divided out, then the negative portions of each cochlear channel are recovered by using the fact that each channel is spectrally limited. Finally, the cochlear filters are inverted by running the filters backwards, and then correcting the resulting spectral slope.

The AGC stage in this cochlear model is controlled by its own output. It is a combination of a multiplicative gain and a simple first-order filter to track the history of the output signal. Since the controlling signal is directly available, the AGC can be inverted by tracking the output history and then dividing instead of multiplying. The performance of this algorithm is described by Naar [10] and will not be addressed here. It is worth noting that AGC inversion becomes more difficult as the level of the input signal is raised, resulting in more compression in the forward path.

The next stage in the inversion process can be done in one of two ways. After AGC inversion, both the positive values of the signal and the spectral extant of the signal are known. Projections onto convex sets [11], in this case defined by the positive values of the detector output and the spectral extant of the cochlear filters, can be used to find the original signal. This is shown in the left half of Figure 3. Alternatively, the spectral projection filter can be combined with the next stage of processing to make the algorithm more efficient. The increased efficiency is due to better match between the spectral projection and the cochlear filterbank, and due to the simplified computations within each iteration. This is shown in the right half of Figure 3. The result is an algorithm that produces nearly perfect results with no iterations at all.

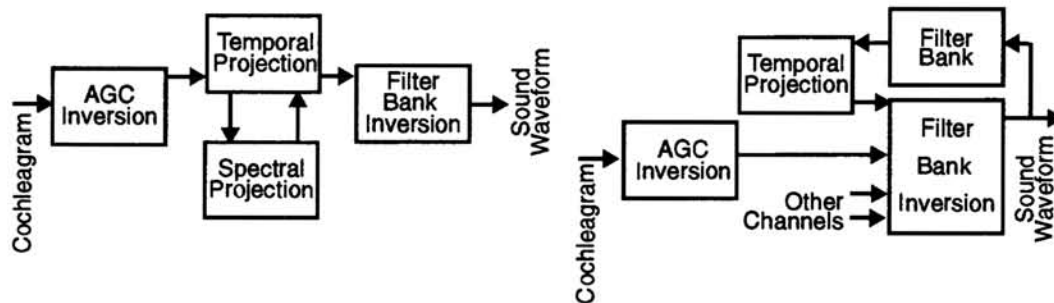

Figure 3. There are two ways to use convex projections to recover the information lost by the detectors. The conventional approach is shown on the left. The right figure shows a more efficient approach where the spectral projection has been combined with the filterbank inversion

Finally, the multiple outputs from the cochlear filterbank are converted back into a single waveform by correcting the phase and summing all channels. In the ideal case, each cochlear channel contains a unique portion of the spectral energy, but with a bit of phase delay and amplitude change. For example, if we run the signal through the same filter the spectral content does not change much but both the phase delay and amplitude change will be doubled. More interestingly, if we run the signal through the filter backwards, the forward and backward phase changes cancel out. After this phase correction, we can sum all channels and get back the original waveform, with a bit of spectral coloration. The spectral coloration or tilt can be fixed with a simple filter. A more efficient approach to correct the spectral tilt is to scale each channel by an appropriate weight before summing, as shown in Figure 4. The result is a perfect reconstruction, over those frequencies where the cochlear filters are non-zero.

Figure 5 shows results from the cochleagram inversion procedure. An impulse is shown on the left, before and after 10 iterations of the HWR inversion (using the algorithm on the right half of Figure 3). With no iterations the result is nearly perfect, except for a bit of noise near the center. The overall curvature of the baseline is due to the fact that informa-

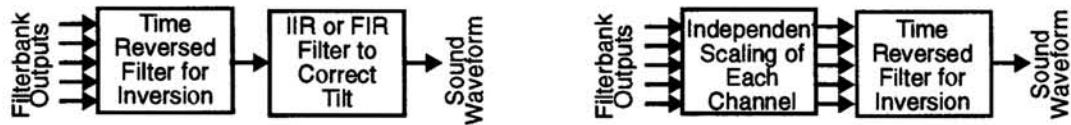

Figure 4. Two approaches are shown here to invert the filterbank. The left diagram shows the normal approach, the right figure shows a more efficient approach where the spectral-tilt filter is converted to a simple multiplication.

tion near DC has been lost as it travels through the auditory system and there is no way to recover it with the information that we have. A more interesting example is shown on the right. Here the word "tap"[1] has been reconstructed, with and without the AGC inversion. With the AGC inversion the result is nearly identical to the original. The auditory system is very sensitive to onsets and quickly adapts to steady state sounds like vowels. It is interesting to compare this to the reconstruction without AGC inversion. Without the AGC, the result is similar to what the ear hears, the onsets are more prominent and the vowels are deemphasized. This is shown in the right half of Figure 5.

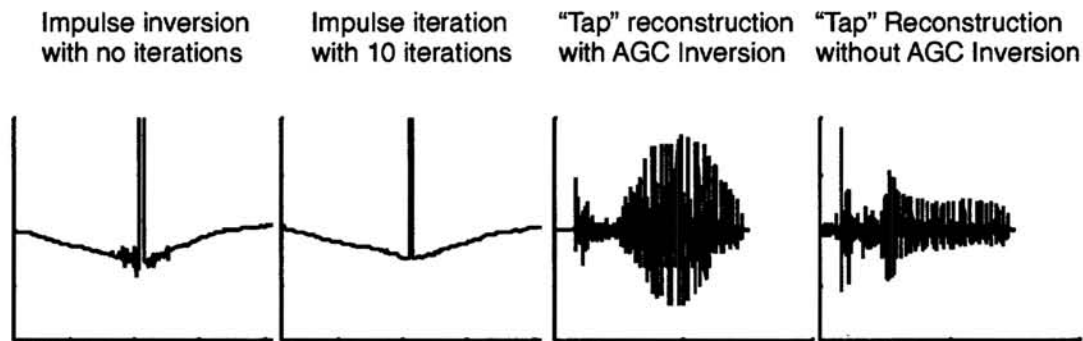

Figure 5. The cochlear reconstructions of an impulse and the word "tap" are shown here. The first and second reconstructions show an impulse reconstruction with and without iterations. The third and fourth waveforms are the word "tap" with and without the AGC inversion.

## 3 CORRELOGRAM INVERSION

The correlogram is an efficient way to capture the short-time periodicities in the auditory signal. Many mechanical measurements of the cochlea have shown that the response is highly non-linear. As the signal level changes there are large variations in the bandwidth and center frequency of the cochlear response. With these kinds of changes, it is difficult to imagine a system that can make sense of the spectral profile. This is especially true for decisions like pitch determination and sound separation.

But through all these changes in the cochlear filters, the timing information in the signal is preserved. The spectral profile, as measured by the cochlea, might change, but the rate of glottal pulses is preserved. Thus I believe the auditory system is based on a representation of sound that makes short-time periodicities apparent. One such representation is the correlogram. The correlogram measures the temporal correlation within each channel, either using FFTs which are most efficient in computer implementations, or neural delay lines much like those found in the binaural system of the owl.

---

1. The syllable "tap", samples 14000 through 17000 of the "train/dr5/fcdf1/ sx106/sx106.adc" utterance on the TIMIT Speech Database, is used in all voiced examples in this paper.

The process of inverting the correlogram is simplified by noting that each autocorrelation is related by the Fourier transform to a power spectrum. By combining many power spectrums into a picture, the result is a spectrogram. This process is shown in Figure 6. In this way, a separate spectrogram is created for each channel. There are known techniques for converting a spectrogram, which has amplitude information but no phase information, back into the original waveform. The process of converting from a spectrogram back into a waveform is described in Section 4. The correlogram inversion process consists of inverting many spectrograms to form an estimate of a cochleagram. The cochleagram is inverted using the techniques described in Section 2.

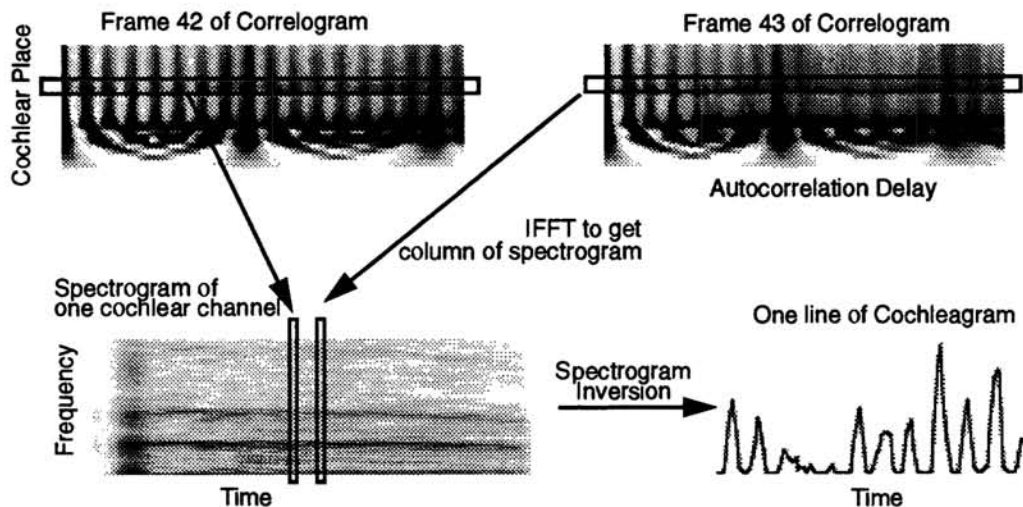

Figure 6. Correlogram inversion is possible by noting that each row of the correlogram contains the same information as a spectrogram of the same row of cochleagram output. By converting the correlogram into many spectrograms, the spectrogram inversion techniques described in Section 4 can be used. The lower horizontal stripe in the spectrogram is due to the narrow passband of the cochlear channel. Half-wave rectification of the cochlear filter output causes the upper horizontal stripes.

One important improvement to the basic method is possible due to the special characteristics of the correlogram. The essence of the spectrogram inversion problem is to recover the phase information that has been thrown away. This is an iterative procedure and would be costly if it had to be performed on each channel. Fortunately, there is quite a bit of overlap between cochlear channels. Thus the phase recovered from one channel can be used to initialize the spectrogram inversion for the next channel. A difficulty with spectrogram inversion is that the absolute phase is lost. By using the phase from one channel to initialize the next, a more consistent set of cochlear channel outputs is recovered.

## 4 SPECTROGRAM INVERSION

While spectrograms are not an accurate model of human perception, an implementation of a correlogram includes the calculation of many spectrograms. Mathematically, an autocorrelation calculation is similar to a spectrogram or short-time power spectrum. One column of a conventional spectrogram is related to an autocorrelation of a portion of the original waveform by a Fourier transform (see Figure 6). Unfortunately, the final representation of both spectrograms and autocorrelations is missing the phase information. The main task of a spectrogram inversion algorithm is to recover a consistent estimate of the missing phase. This process is not magical, it can only recover a signal that has the same magnitude spectrum as the original spectrogram. But the consistency constraint on the time evolution of the signal power spectrum also constrains the time evolution of the spectral phase.

The basic procedure in spectrogram inversion [12] consists of iterating between the time and the frequency domains. Starting from the frequency domain, the magnitude but not the phase is known. As an initial guess, any phase value can be used. The individual power spectra are inverse Fourier transformed and then summed to arrive at a single waveform. If the original spectrogram used overlapping windows of data, the information from adjacent windows either constructively or destructively interferes to estimate a waveform. A spectrogram of this new data is calculated, and the phase is now retained. We know the original magnitude was correct. Thus we can estimate a better spectrogram by combining the original magnitude information with the new phase information. It can be shown that each iteration will reduce the error.

Figure 7 shows an outline of steps that can be used to improve the consistency of phase estimates during the first iteration. As each portion of the waveform is added to the estimated signal, it is possible to add a linear phase so that each waveform lines up with the proceedings segments. The algorithm described in the paragraph above assumes an initial phase of zero. A more likely phase guess is to choose a phase that is consistent with the existing data. The result with no iterations is a waveform that is often closer to the original than that calculated assuming zero initial phase and ten iterations.

The total computational cost is minimized by combining these improvements with the initial phase estimates from adjacent channels of the correlogram. Thus when inverting the first channel of the correlogram, a cross-correlation is used to pick the initial phase and a few more iterations insure a consistent result. After the first channel, the phase of the proceeding channel is used to initialize the spectrogram inversion and only a few iterations are necessary to fine tune the waveform.

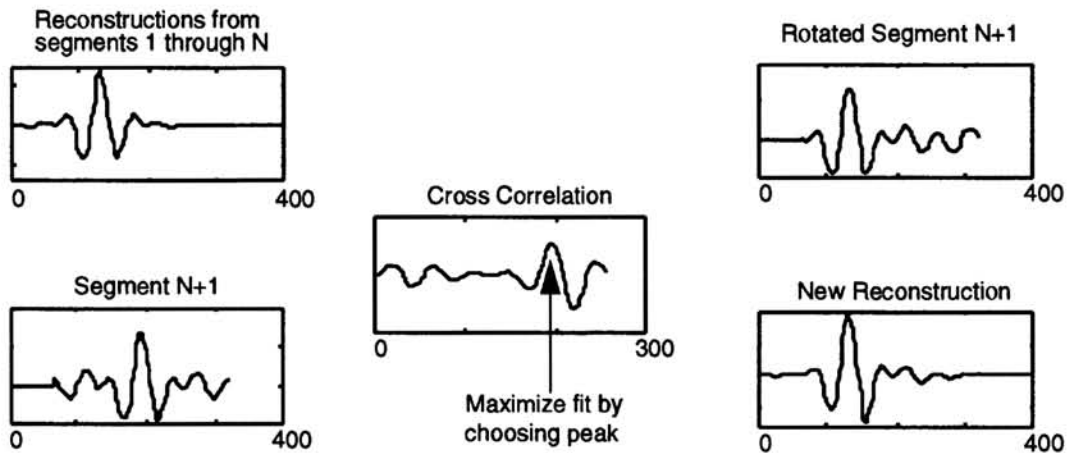

Figure 7. A procedure for adjusting the phase of new segments when inverting a spectrogram is shown above. As each new segment (bottom left) is converted from a power spectrum into a waveform, a linear phase is added to maximize the fit with the existing segments (top left.) The amount of rotation is determined by a cross correlation (middle). Adding the new segment with the proper rotation (top right) produces the new waveform (bottom right.)

## 5 PUTTING IT TOGETHER

This paper has described two steps to convert a correlogram into a sound. These steps are detailed below:

1) For each row of the correlogram:
   a) Convert the autocorrelation data into power spectrum (Section 3).
   b) Use spectrogram inversion (Section 4) to convert the spectrograms into an estimate of cochlear channel output.
   c) Assemble the results of spectrogram inversion into an estimate of the cochleagram.

2) Invert the cochleagram using the techniques described in Section 2.

This process is diagrammed in Figures 1 and 6.

## 6 RESULTS

Figure 8 shows the results of the complete reconstruction process for a 200Hz impulse train and the word "tap." In both cases, no iterations were performed for either the spectrogram or filterbank inversion. More iterations reduce the spectral error, but do not make the graphs look better or change the perceptual quality much. It is worth noting that the "tap" reconstruction from a correlogram looks similar to the cochleagram reconstruction without the AGC (see Figure 5.) Reducing the level of the input signal, thus reducing the amount of compression performed by the AGC, results in a correlogram reconstruction similar to the original waveform.

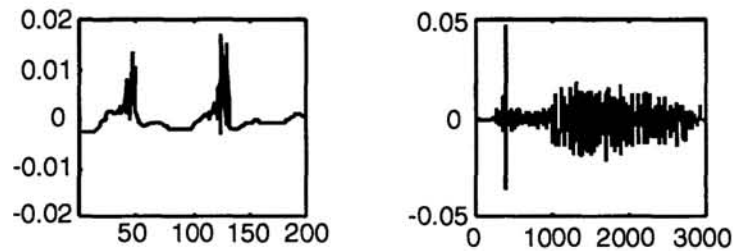

Figure 8. Reconstructions from the correlogram representation of an impulse train and the word "tap" are shown above. Reducing the input signal level, thus minimizing the effect of errors when inverting the AGC, produces results identical to the original "tap."

It is important to note that the algorithms described in this paper are designed to minimize the error in the mean-square sense. This is a convenient mathematical definition, but it doesn't always correlate with human perception. A trivial example of this is possible by comparing a waveform and a copy of the waveform delayed by 10ms. Using the mean-squared error, the numerical error is very high yet the two waveforms are perceptually equivalent. Despite this, the results of these algorithms based on mean-square error do sound good.

## 7 CONCLUSIONS

This paper has described several techniques that allow several stages of an auditory model to be converted back into sound. By converting each row of the correlogram into a spectrogram, the spectrogram inversion techniques of Section 4 can be used. The special characteristics of a correlogram described in Section 3 are used to make the calculation more efficient. Finally, the cochlear filterbank can be inverted to recover the original waveform. The results are waveforms, perceptually identical to the original waveforms.

These techniques will be especially useful as part of a sound separation system. I do not believe that our auditory system resynthesizes partial waveforms from the auditory scene. Yet, all research systems generate separated sounds so that we can more easily perceive their success. More work is still needed to fine-tune these algorithm and to investigate the ability to reconstruct sounds from partial correlograms.

## Acknowledgments

I am grateful for the inspiration provided by Frank Cooper's work in the early 1950's on pattern playback[13][14]. His work demonstrated that it was possible to convert a spectrogram, painted onto clear plastic, into sound.

This work in this paper was performed with Daniel Naar and Richard F. Lyon. We are grateful for the help we have received from Richard Duda (San Jose State), Shihab Shamma (U. of Maryland), Jim Boyles (The MathWorks) and Michele Covell (Interval Research).

## References

[1] Malcolm Slaney, D. Naar, R. F. Lyon, "Auditory model inversion for sound separation," *Proc. of IEEE ICASSP*, Volume II, pp. 77-80, 1994.
[2] M. Slaney and R. F. Lyon, "On the importance of time—A temporal representation of sound," in *Visual Representations of Speech Signals*, eds. M. Cooke, S. Beet, and M. Crawford, J. Wiley and Sons, Sussex, England, 1993.
[3] R. F. Lyon, "A computational model of binaural localization and separation," *Proc. of IEEE ICASSP*, 1148-1151, 1983.
[4] M. Weintraub, "The GRASP sound separation system," *Proc. of IEEE ICASSP*, pp. 18A.6.1-18A.6.4, 1984.
[5] D. Hermes, "Pitch analysis," in *Visual Representations of Speech Signals*, eds. M. Cooke, S. Beet, and M. Crawford, J. Wiley and Sons, Sussex, England, 1993.
[6] N. Suga, "Cortical computational maps for auditory imaging," *Neural Networks*, 3, 3-21, 1990.
[7] T. Irino, H. Kawahara, "Signal reconstruction from modified auditory wavelet transform," *IEEE Trans. on Signal Processing*, 41, 3549-3554, Dec. 1993.
[8] X. Yang, K. Wang, and S. Shamma, "Auditory representations of acoustic signals," *IEEE Trans. on Information Theory*, 38, 824-839, 1992.
[9] R. F. Lyon, "A computational model of filtering, detection, and compression in the cochlea," *Proc. of the IEEE ICASSP*, 1282-1285, 1982.
[10] D. Naar, "Sound resynthesis from a correlogram," San Jose State University, Department of Electrical Engineering, Technical Report #3, May 1993.
[11] R. W. Papoulis, "A new algorithm in spectral analysis and band-limited extrapolation," *IEEE Trans. Circuits Sys.*, vol. 22, 735, 1975.
[12] D. Griffin and J. Lim, "Signal estimation from modified short-time Fourier transform," *IEEE Trans. on Acoustics, Speech, and Signal Processing*, 32, 236-242, 1984.
[13] F. S. Cooper, "Some Instrumental Aids to Research on Speech," *Report on the Fourth Annual Round Table Meeting on Linguistics and Language Teaching*, Georgetown University Press, 46-53, 1953.
[14] F. S. Cooper, "Acoustics in human communications: Evolving ideas about the nature of speech," *J. Acoust. Soc. Am.*, 68(1), 18-21, July 1980.